# Temporal and Cross-Subject Probabilistic Models for fMRI Prediction Tasks

**Alexis Battle**  **Gal Chechik**  **Daphne Koller**
Department of Computer Science
Stanford University
Stanford, CA 94305-9010
{ajbattle,gal,koller}@cs.stanford.edu

## Abstract

We present a probabilistic model applied to the fMRI video rating prediction task of the Pittsburgh Brain Activity Interpretation Competition (PBAIC) [2]. Our goal is to predict a time series of subjective, semantic ratings of a movie given functional MRI data acquired during viewing by three subjects. Our method uses conditionally trained Gaussian Markov random fields, which model both the relationships between the subjects' fMRI voxel measurements and the ratings, as well as the dependencies of the ratings across time steps and between subjects. We also employed non-traditional methods for feature selection and regularization that exploit the spatial structure of voxel activity in the brain. The model displayed good performance in predicting the scored ratings for the three subjects in test data sets, and a variant of this model was the third place entrant to the 2006 PBAIC.

## 1 Introduction

In functional Magnetic Resonance Imaging, or *fMRI*, an MR scanner measures a physiological signal known to be correlated with neural activity, the blood-oxygenation-level dependent (BOLD) signal [12]. Functional scans can be taken during a task of interest, such as the subject viewing images or reading text, thus providing a glimpse of how brain activity changes in response to certain stimuli and tasks. An fMRI session produces scans of the brain volume across time, obtaining BOLD measurements from thousands of small sub-volumes, or *voxels* at each time step.

Much of the current fMRI research focuses on the goal of identifying brain regions activated in response to some task or stimulus (e.g., [7]). The fMRI signal is typically averaged over many repeated stimulus presentations, multiple time points and even different subjects, in order to find brain regions with statistically significant response. However, in recent years, there has been growing interest in an alternative task, whose goal is to develop models which predict stimuli from functional data, in effect demonstrating the ability to 'read' information from the scans. For instance, Tong et al. [9] demonstrated the ability to predict the orientation of edges in a subject's visual field from functional scans of visual cortex, and Mitchell et al. [13] successfully applied machine learning techniques to a predict a variety of stimuli, such as the semantic category of words presented to a subject. Such prediction work has demonstrated that, despite the relatively low spatial resolution of fMRI, functional data contains surprisingly reliable and detailed signal [9, 6, 13], even on time scales as short as a few seconds. Going beyond identifying the location of responsive regions, these models begin to demonstrate *how* the brain encodes states and stimuli [3], often capturing distributed patterns of activation across multiple brain regions simultaneously. This line of research could also eventually provide a mechanism for accurately tracking cognitive processes in a non-invasive way.

Another recent innovation is the use of long and rich stimuli in fMRI experiments, such as a commercial movie [8], rather than the traditional controlled, repeating simple stimuli. These experiments

present more difficulty in analysis, but more closely mirror natural stimulation of the brain, which may evoke different brain activity patterns from traditional experiments.

The recent *Pittsburgh Brain Activity Interpretation Competition* [2] (PBAIC), featured *both* the use of complex stimuli and a prediction task, presenting a unique data set for predicting subjective experiences given functional MRI sessions. Functional scans from three subjects were taken while the subjects watched three video segments. Thus, during the scan, subjects were exposed to rich stimuli including rapidly changing images of people, meaningful sounds such as dialog and music, and even emotional stimuli, all overlapping in time. Each subject also re-viewed each movie multiple times, to rate over a dozen characteristics of the videos over time, such as **Amusement**, presence of **Faces** or **Body Parts**, **Language**, and **Music**. Given this data set, the goal was to predict these real-valued subjective ratings for each subject based only on the fMRI scans.

In this paper, we present an approach to the PBAIC problem, based on the application of machine learning methods within the framework of probabilistic graphical models. The structured probabilistic framework allowed us to represent many relevant relationships in the data, including evolution of subjective ratings over time, the likelihood of different subjects rating experiences similarly, and of course the relationship between voxels and ratings. We also explored novel feature selection methods, which exploit the spatial characteristics of brain activity. In particular, we incorporate a bias in favor of jointly selecting nearby voxels.

We demonstrate the performance of our model by training from a subset of the movie sessions and predicting ratings for held out movies. An earlier variant of our model was the third place entrant to the 2006 PBAIC out of forty entries. We demonstrated very good performance in predicting many of the ratings, suggesting that probabilistic modeling for the fMRI domain is a promising approach. An analysis of our learned models, in particular our feature selection results, also provides some insight into the regions of the brain activated by different stimuli and states.

## 2 Probabilistic Model

Our system for prediction from fMRI data is based on a dynamic, undirected graphical probabilistic model, which defines a large structured conditional Gaussian over time and subjects. The backbone of the model is a conditional linear Gaussian model, capturing the dependence of ratings on voxel measurements. We then extend the basic model to incorporate dependencies between labels across time and between subjects.

The variables in our model are voxel activations and ratings. For each subject $s$ and each time point $t$, we have a collection of ratings $R_s(\cdot, t)$, with $R_s(j, t)$ representing the $j$th rating type (for instance **Language**) for $s$ at time $t$. Note that the rating sequences given by the subjects are actually convolved with a standard hemodynamic response function before use, to account for the delay inherent in the BOLD signal response [4]. For each $s$ and $t$, we also have the voxel activities $V_s(\cdot, t)$. Both voxels and ratings are continuous variables. For mathematical convenience, we recenter the data such that all variables (ratings and voxels) have mean 0.

Each rating $R_s(j, t)$ is modeled as a linear Gaussian random variable, dependent only on voxels from that subject's brain as features. We can express $R_s(j, t) \sim \mathcal{N}(\boldsymbol{w}_s(j)^T V_s(\cdot, t), \sigma_s^2)$. We assume that the dependence of the rating on the voxels is time-invariant, so that the same parameters $\boldsymbol{w}_s(j)$ and $\sigma_s$ are used for every time point. Importantly, however, each rating should not depend on *all* of the subject's voxels, as this is neither biologically likely nor statistically plausible given the large number of voxels. In Sec. 3.1 we explore a variety of feature selection and regularization methods relevant to this problem.

The linear regression model forms a component in a larger model that accounts for dependencies among labels across time and across subjects. This model takes the form of a (dynamic) Gaussian Markov Random Field (GMRF) [15, 11]. A GMRF is an undirected graphical probabilistic model that expresses a multi-dimensional joint Gaussian distribution in a reduced parameter space by making use of conditional independencies. Specifically, we employ a standard representation of a GMRF derived from the inverse covariance matrix, or *precision matrix* $Q = \Sigma^{-1}$ of the underlying Gaussian distribution: For $\boldsymbol{X} = (X_1, \ldots, X_n)$, a zero-mean joint Gaussian distribution over $\boldsymbol{X}$ can be written as $P(\boldsymbol{X}) \propto \exp(-\frac{1}{2}\boldsymbol{X}^T Q \boldsymbol{X})$. The precision matrix maps directly to a Markov network

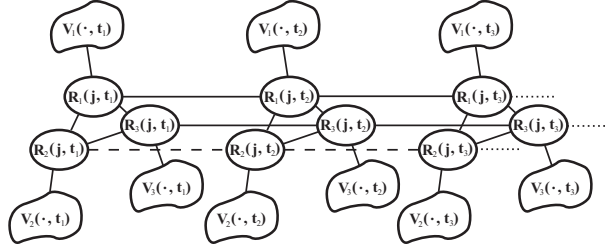

Figure 1: GMRF model for one rating, $R_{\cdot}(j,t)$, over three subjects and three time steps.

representation, as $Q(i,j) = 0$ exactly when $X_i$ is independent of $X_j$ given the remaining variables, corresponding to the absence of an edge between $X_i$ and $X_j$ in the Markov network.

In our setting, we want to express a conditional linear Gaussian of the ratings given the voxels. A distribution $P(\boldsymbol{X} \mid \boldsymbol{Y})$ can also be parametrized using the joint precision matrix:

$$P(\boldsymbol{X} \mid \boldsymbol{Y}) = \frac{1}{Z(Y)} \prod_i \exp\left(-\frac{1}{2}Q_{XX}(i,i)X_i^2\right) \prod_{i,j \in E_X} \exp\left(-Q_{XX}(i,j)X_iX_j\right)$$
$$\prod_{i,k \in E_Y} \exp\left(-Q_{XY}(i,k)X_iY_k\right),$$

where $E_X$ is the set of edges between nodes in $\boldsymbol{X}$, and $E_Y$ represents edges from $\boldsymbol{Y}$ to $\boldsymbol{X}$.

Our particular GMRF is a joint probabilistic model that encompasses, for a particular rating type $j$, the value of the rating $R_s(j,t)$ for all of the subjects $s$ across all time points $t$. Our temporal model assumes a stationary distribution, so that both node and edge potentials are invariant across time. This means that several entries in the full precision matrix $Q$ are tied to a single free parameter. We will treat each rating type separately. Thus, the variables in the model are: all of the voxel measurements $V_s(l,t)$, for all $s,t$ and voxels $l$ selected to be relevant to rating $j$; and all of the ratings $R_s(j,t)$ (for all $s,t$). As we discuss below, the model is trained conditionally, and therefore encodes a joint distribution over the rating variables, *conditional* on all of the voxel measurements. Thus, there will be no free parameters corresponding to the voxel nodes due to the use of a conditional model, while rating nodes $R_s(j,\cdot)$ have an associated node potential parameter $Q_{node}(s,j)$.

Each rating node $R_s(j,t)$ has edges connecting it to a subset of relevant voxels from $V_s(\cdot,t)$ at the same time slice. The set of voxels can vary for different ratings or subjects, but is consistent across time. The precision matrix entry $Q_{voxel}(s,j,v)$ parametrizes the edge from voxel $v$ to rating $j$. To encode the dependencies between the rating at different time points, our dynamic model includes edges between each rating $R_s(j,t)$ and the previous and following ratings, $R_s(j,t-1)$ and $R_s(j,t+1)$. The corresponding edge parameters are $Q_{time}(s,j)$. We also use the GMRF to encode the dependencies between the ratings of different subjects, in a way that does not assume that the subjects gave identical ratings, by introducing appropriate edges in the model. Thus, we also have an edge between $R_s(j,t)$ and $R_{s'}(j,t)$ for all subject pairs $s,s'$, parametrized by $Q_{subj}(s,s',j)$.

Overall, our model encodes the following conditional distribution:

$$P(R_{\cdot}(j,\cdot) \mid V_{\cdot}(\cdot,\cdot)) =$$
$$\frac{1}{Z(V_{\cdot}(\cdot,\cdot))} \prod_{s,t} \exp\left(-\frac{1}{2}Q_{node}(s,j)R_s(j,t)^2\right) \prod_{t,s,s'} \exp\left(-Q_{subj}(s,s',j)R_s(j,t)R_{s'}(j,t)\right)$$
$$\prod_{s,t,t+1} \exp\left(-Q_{time}(s,j)R_s(j,t)R_s(j,t+1)\right) \prod_{s,l,t} \exp\left(-Q_{voxel}(s,j,l)R_s(j,t)V_s(l,t)\right). \quad (1)$$

## 3   Learning and Prediction

We learn the parameters of the model above from a data set consisting of all of the voxels and all the subjective ratings for all three subjects. We train the parameters discriminatively [10], to maximize the conditional likelihood of the observed ratings given the observed voxel measurements,

as specified in Eq. (1). Conditional training is appropriate in our setting, as our task is precisely to predict the ratings given the voxels; importantly, this form of training allows us to avoid modeling the highly noisy, high-dimensional voxel activation distribution. We split parameter learning into two phases, first learning the dependence of ratings on voxels, and then learning the parameters between rating nodes. The entire joint precision matrix over all voxels and ratings would be prohibitively large for our learning procedure, and this approximation was computationally much more efficient. In the first phase, we learn linear models to predict each rating given only the voxel activations. We then modify our graph, replacing the very large set of voxel nodes with a new, much smaller set of nodes representing the linear combinations of the voxel activations which we just learned. Using the reduced graph, we learn a much smaller precision matrix. We describe each of these steps below.

## 3.1 From Voxels to Ratings

To learn the dependencies of ratings on voxels for a single subject $s$, we find parameters $\boldsymbol{w}_s(j)$, using linear regression, which optimize

$$P(R_s(j, \cdot) \mid V_s(\cdot, \cdot)) \propto \prod_t \exp\left(-\frac{1}{2\sigma_s^2}(R_s(j, t) - \boldsymbol{w}_s(j)^T V_s(\cdot, t))^2\right). \tag{2}$$

However, to deal with the high dimensionality of the feature space relative to the number of training instances, we utilize feature selection; we also introduce regularization terms into the objective that can be viewed as a spatially-based prior over $\boldsymbol{w}_s(j)$. First, we reduce the number of voxels involved in the objective for each rating using a simple feature selection method — we compute the Pearson correlation coefficient for each voxel and each rating, and select the most highly correlated features. The number of voxels to select is a setting which we tuned, for each rating type individually, using five-fold cross-validation on the training set. We chose to use the same number of voxels across subjects, which is more restrictive but increases the amount of data available for cross-validation.

Even following this feature selection process, we often still have a large number (perhaps hundreds) of relevant voxels as features, and these features are quite noisy. We therefore employ additional regularization over the parameters associated with these voxels. We explored both $L_2$ (ridge) and $L_1$ (Lasso) regularization, corresponding to a Gaussian and a Laplacian prior respectively. Introducing both types of regularization, we end up with a log-likelihood objective of the form:

$$\sum_t (R_s(j, t) - \boldsymbol{w}_s(j)^T V_s(\cdot, t))^2 + \alpha \sum_i \boldsymbol{w}_s(j, i)^2 + \beta \sum_i |\boldsymbol{w}_s(j, i)| \tag{3}$$

Finally, we introduce a novel form of regularization, intended to model spatial regularities. Brain activity associated with some types of stimuli, such as language, is believed to be localized to some number of coherent regions, each of which contains multiple activated voxels. We therefore want to bias our feature selection process in favor of selecting multiple voxels that are nearby in space; more precisely, we would prefer to select a voxel which is in the vicinity of other correlated voxels, over a more strongly correlated voxel which is isolated in the brain, as the latter is more likely to result from noise. We therefore define a robust "hinge-loss"-like distance function for voxels. Letting $\|v_i - v_k\|_2$ denote the Euclidean distance between voxels $v_i$ and $v_k$ in the brain, we define:

$$D(i, k) = \begin{cases} 1 & \text{if } \|v_i - v_k\|_2 < d_{min}, \\ 0 & \text{if } \|v_i - v_k\|_2 > d_{max}, \\ \frac{d_{max} - \|v_i - v_k\|_2}{d_{max} - d_{min}} & \text{otherwise.} \end{cases}$$

We now introduce an additional regularization term

$$-\lambda \sum_{ik} |\boldsymbol{w}_s(j, i)| D(i, k) |\boldsymbol{w}_s(j, k)|$$

into the objective Eq. (3). This term can offset the $L_1$ term by co-activating voxels that are spatially nearby. Thus, it encourages, but does not force, co-selection of nearby voxels. Note that this regularization term is applied to the absolute values of the voxel weights, hence allowing nearby voxels to have opposite effects on the rating; we do observe such cases in our learned model. Note that, according to our definition, the spatial prior uses simple Euclidean distance in the brain. This is clearly too simplistic, as it ignores the structure of the brain, particularly the complex folding of the cortex. A promising extension of this idea would be to apply a geodesic version of distance instead, measuring distance over gray matter only.

## 3.2 Training the Joint Model

We now describe the use of regression parameters, as learned in Sec. 3.1, to reduce the size of our joint precision matrix, and learn the final parameters including the inter-rating edge weights.

Given $\boldsymbol{w}_s(j)$, which we consider the optimal linear combination of $V_s(\cdot, j)$ for predicting $R_s(j)$, we remove the voxel nodes $V_s(\cdot, t)$ from our model, and introduce new 'summary' nodes $U_j(t) = \boldsymbol{w}_s(j)^T V_s(\cdot, t)$. Now, instead of finding $Q_{voxel}(s, j, v)$ parameters for every voxel $v$ individually, we only have to find a single parameter $Q_u(s, j)$. Given the structure of our original linear Gaussian model, there is a direct relationship between optimization in the reduced formulation and optimizing using the original formulation. Assuming $\boldsymbol{w}_s(j)$ is the optimal set of regression parameters, the optimal $Q_{voxel}(s, j, l)$ in the full form would be proportional to $Q_u(s, j)\boldsymbol{w}_s(j, l)$, optimized in the reduced form. This does *not* guarantee that our two phase learning results in globally optimal parameter settings, but simply that given $\boldsymbol{w}_s(j)$, the reduction described is valid.

The joint optimization of $Q_u(s, j)$, $Q_{node}(s, j)$, $Q_{time}(s, j)$, and $Q_{subj}(s, s', j)$ is performed according to the reduced conditional likelihood. The reduced form of Eq. (1) simply replaces the final terms containing $Q_{voxel}(s, j, )$ with:

$$\prod_{s,t} \exp\left(-Q_u(s, j)R_s(j, t)U_j(t)\right). \tag{4}$$

The final objective is computationally feasible due to the reduced parameter space. The log likelihood is a convex function of all our parameters, with the final joint precision matrix constrained to be positive semi-definite to ensure a legal Gaussian distribution. Thus, we can solve the problem with semi-definite programming using a standard convex optimization package [1]. Last, we combine all learned parameters from both steps, repeated across time steps, for the final joint model.

## 3.3 Prediction

Prediction of unseen ratings given new fMRI scans can be obtained through probabilistic inference on the models learned for each rating type. We incorporate the observed voxel data from all three subjects as observed values in our GMRF, which induces a Gaussian posterior over the joint set of ratings. We only need to predict the most likely assignment to ratings, which is the mean (or mode) of this Gaussian posterior. The mean can be easily computed using coordinate ascent over the log likelihood of our joint Gaussian model. More precisely, we iterate over the nodes (recall there is one node for each subject at each time step), and update its mean to the most likely value given the current estimated means of its neighbors in the GMRF. Let $Q_{RR}$ be the joint precision matrix, over all nodes over time and subject, constructed from $Q_u(\cdot, \cdot)$, $Q_{time}(\cdot, \cdot)$, $Q_{subj}(\cdot, \cdot, \cdot)$, and $Q_{node}(\cdot, \cdot)$. Then for each node $k$ and neighbors $N_k$ according to the graph structure of our GMRF, we update $\mu_k \Leftarrow -\sum_{j \in N_k} \mu_j Q_{RR}(k, j)$. As the objective is convex, this process is guaranteed to converge to the mode of the posterior Gaussian, providing the most likely ratings for all subjects, at all time points, given the functional data from scans during a new movie.

## 4 Experimental Results

As described, the fMRI data collected for the PBAIC included fMRI scans of three different subjects, and three sessions each. In each of the sessions, a subject viewed a movie approximately 20 minutes in length, constructed from clips of the *Home Improvement* sitcom. All three subjects watched the same movies — referred to as *Movie1*, *Movie2* and *Movie3*. The scans produced volumes with approximately $30,000$ brain voxels, each approximately 3.28mm by 3.28mm by 3.5mm, with one volume produced every 1.75 seconds. Subsequently, the subject watched the movie again multiple times (not during an fMRI session), and rated a variety of characteristics at time intervals corresponding to the fMRI volume rate. Before use in prediction, the rating sequences were convolved with a standard hemodynamic response function [4]. The core ratings used in the competition were **Amusement**, **Attention**, **Arousal**, **Body Parts**, **Environmental Sounds**, **Faces**, **Food**, **Language**, **Laughter**, **Motion**, **Music**, **Sadness**, and **Tools**. Since the ratings are continuous values, competition scoring was based on the correlation (for frames where the movie is playing) of predicted

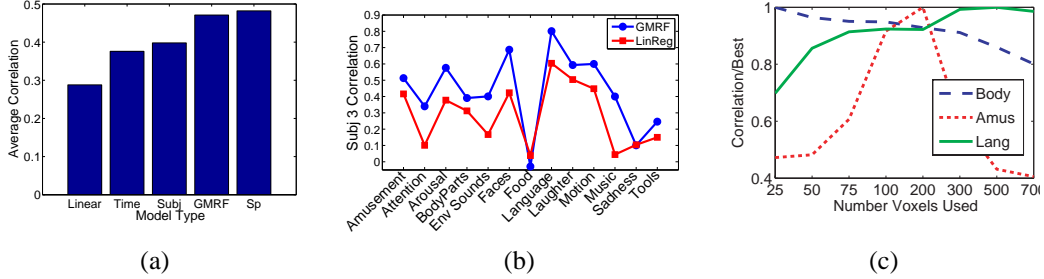

Figure 2: (a) Average correlation of predicted ratings and true ratings, for simple models, the full GMRF, finally including the spatial (Sp) prior. (b) Correlations for individual ratings, for subject 3. (c) Effect of varying the number of voxels used for Language, Amusement, and BodyParts.

ratings with true ratings, across rating types and all subjects, combined using a $z'$-transform. For consistency, we adhere to the use of correlation as our performance metric.

To train our model, we use the fMRI measurements along with all ratings from all subjects' sessions for some set of movies, holding out other movies for testing. We chose to use an entire held out movie session because the additional variance between fMRI sessions is an important aspect of the prediction task. The training set is used both to learn the model parameters and for the cross-validation step used to select regularization settings. The learned model is then used to predict ratings for the held out test movies for all subjects, from fMRI data alone. Our GMRF model shows significant improvement over simpler models on the prediction task, and a version of this model was used in our submission to the PBAIC. We also evaluate the results of our feature selection steps, examining which regions of the brain are used for each rating prediction.

## 4.1 Rating Prediction

For our own evaluation outside of the competition, given that we did not have access to *Movie3* ratings for testing, we trained a full model using functional data and ratings from the three subjects viewing *Movie1*, and then made predictions using the scans from all subjects for *Movie2*. The predictions made by the dynamic GMRF model were highly correlated with the true ratings. The best overall average correlation achieved for held out *Movie2* was $0.482$. For all subjects, the correlation for both **Language** and **Faces** was above $0.7$, and we achieved correlations of above $0.5$ on 19 of the 39 core tasks (three subjects time 13 ratings).

To evaluate the contribution of various components of our model, we also tested simpler versions, beginning with a regularized linear regression model. We also constructed two simplified versions of our GMRF, one which includes edges between subjects but not time interactions, and conversely one which includes time interactions but removed subject edges. Finally, we tested our full GMRF model, plus our GMRF model along with the spatial prior. As shown in Fig. 2(a), both the time dependencies and the cross-subject interactions help greatly over the linear regression model. The final combined model, which includes both time and subject edges, demonstrates significant improvement over including either alone. We also see that the addition of a spatial prior (using cross-validation to select which ratings to apply it to), results in a small additional improvement, which we explore further in Sec. 4.2. Performance on each of the rating types individually is shown in Fig. 2(b) for subject 3, for both linear regression and our GMRF. One interesting note is that the relative ordering of rating type accuracy for the different models is surprisingly consistent.

As mentioned, we submitted the third place entry to the 2006 PBAIC. For the competition, we used our joint GMRF model, but had not developed the spatial prior presented here. We trained the model using data from *Movie1* and *Movie2* and the corresponding ratings from all three subjects. We submitted predictions for the unseen *Movie3* predictions. Around 40 groups made final submissions. Our final score in the competition was $0.493$, whereas $80\%$ of the entries fell below $0.4000$. The first place group, Olivetti et al. [14], employed recurrent neural networks with mutual information based feature selection, scored $0.515$. The second group, scoring $0.509$, was Chigirev et al. [5] — they applied regularized linear models with smoothing across time, spatially nearby voxels and

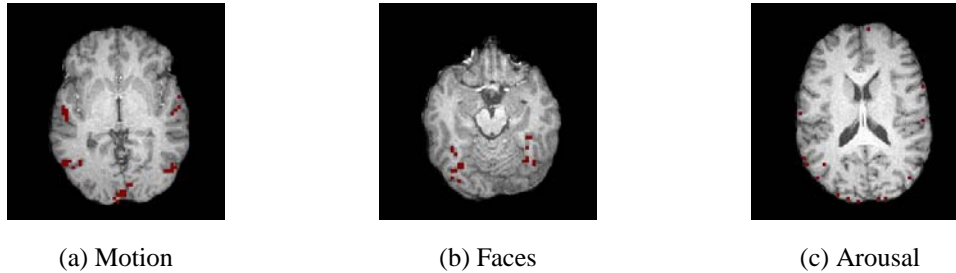

(a) Motion          (b) Faces          (c) Arousal

Figure 3: Voxels selected for various rating predictions, all for Subject 3.

averaging across subjects. Some groups employed machine learning techniques such as Support Vector Regression, while others focused on defined Regions of Interest as features in prediction.

## 4.2 Voxel Selection and Regularization

We also examined the results of feature selection and regularization, looking at the location of voxels used for each rating, and the differences resulting from various techniques. Starting with the approximately $30,000$ brain voxels per subject, we apply our feature selection techniques, using cross-validation on training sessions to determine the number of voxels used to predict each rating. The optimal number did vary significantly by rating, as the graph of performance in Fig. 2(c) demonstrates. For instance, a small voxel set (less than 100) performs well for the **Body Parts** rating, while the **Language** rating does well with several hundred voxels, and **Amusement** uses an intermediate number. This may reflect the actual size and number of brain regions activated by such stimuli, but likely also reflects voxel noise and the difficulty of the individual predictions.

Visualization demonstrates that our selected voxels often occur in regions known to be responsive to relevant stimuli. For instance, voxels selected for **Motion** in all subjects include voxels in cortical areas known to respond to motion in the visual field (Fig. 3(a)). Likewise, many voxels selected for **Language** occur in regions linked to language processing (Fig. 4(b)). However, many other voxels were not from expected brain regions, attributable in part to noise in the data, but also due to the intermixed and correlated stimuli in the videos. For instance, the ratings **Language** and **Faces** for subject 1 in *Movie1* have correlation $0.68$, and we observed that the voxels selected for **Faces** and **Language** overlapped significantly. Voxels in the language centers of the brain improve the prediction of **Faces** since the two stimuli are causally related, but it might be preferable to capture this correlation by adding edges between the rating nodes of our GMRF. Interestingly, there was some consistency in voxel selection between subjects, even though our model did not incorporate cross-subject voxel selection. Comparing **Faces** voxels for Subject 3 Fig. 3(b), to voxels for Subject 2 Fig. 4(a), we see that the respective voxels do come from similar regions. This provides further evidence that the feature selection methods are finding real patterns in the fMRI data.

Finally, we discuss the results of applying our spatial prior. We added the prior for ratings which it improved in cross-validation trials for *all* subjects — **Motion**, **Language**, and **Faces**. Comparison of the voxels selected with and without our spatial prior reveal that it does result in more spatially coherent groups of voxels. Note the *total number* of voxels selected does not rise in general. As shown in Fig. 4(a), the voxels for **Faces** for subject 3 include a relevant group of voxels even without the prior, but including the spatial prior results in inluding additional voxels voxels near this region. Similar results for **Language** are shown for subject 1. **Arousal** prediction was actually hurt by including the spatial prior, and looking at the voxels selected for subject 2 for **Arousal** Fig. 3(c), we see that there is almost no spatial grouping originally, so perhaps here the spatial prior is implausible.

## 5  Discussion

This work, and the other PBAIC entries, demonstrated that a wide range of subjective experiences can be predicted from fMRI data collected during subjects' exposure to rich stimuli. Our probabilistic model in particular demonstrated the value of time-series and multi-subject data, as the use of edges representing correlations across time and correlations between subjects each improved the

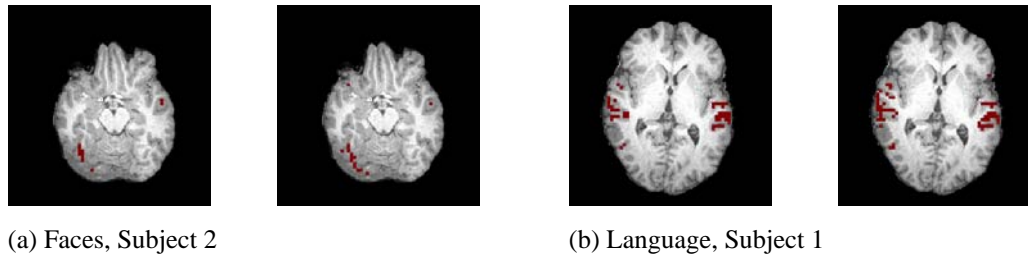

           (a) Faces, Subject 2                        (b) Language, Subject 1

Figure 4: Effect of applying the spatial prior — each left image is without, right is with prior applied.

accuracy of our predictions significantly. Further, while voxels are very noisy, with appropriate regularization and the use of a spatially-based prior, reliable prediction was possible using individual voxels as features. Although voxels were selected from the whole brain, many of the voxels selected as features in our model were located in brain regions known to be activated by relevant stimuli.

One natural extension to our work would include the addition of interactions between distinct rating types, such as Language and Faces, which are likely to be correlated. This may improve predictions, and could also result in more targeted voxel selection for each rating. More broadly, though, the PBAIC experiments provided an extremely rich data set, including complex spatial and temporal interactions among brain voxels and among features of the stimuli. There are many aspects of this data we have yet to explore, including modeling the relationships between the voxels themselves across time, perhaps identifying interesting cascading patterns of voxel activity. Another interesting direction would be to determine which temporal aspects of the semantic ratings are best encoded by brain activity — for instance it is possible that brain activity may respond more strongly to *changes* in some stimuli rather than simply stimulus presence. Such investigations could provide further insight into brain activity in response to complex stimuli in addition to improving our ability to make accurate predictions from fMRI data.

## Acknowledgments

This work was supported by NSF grant DBI-0345474.

## References

[1] Cvx matlab software. http://www.stanford.edu/ boyd/cvx/.

[2] Pittsburgh brain activity interpretation competition inferring experience based cognition from fmri. http://www.ebc.pitt.edu/competition.html.

[3] What's on your mind? *Nature Neuroscience*, 9(981), 2006.

[4] G. M. Boynton, S. A. Engel, G. H. Glover, and D. J. Heeger. Linear systems analysis of functional magnetic resonance imaging in human v1. *J. Neurosci*, 16:4207–4221, 1996.

[5] D. Chigirev, G. Stephens, and T. P. E. team. Predicting base features with supervoxels. Abstract presented, 12th HBM meeting, Florence, Italy, 2006.

[6] D. D. Cox and R. L. Savoya. Functional magnetic resonance imaging (fmri) brain reading: detecting and classifying distributed patterns of fmri activity in human visual cortex. *NeuroImage*, 19:261–270, 2003.

[7] K. J. Friston, A. P. Holmes, K. J. Worsley, J. P. Poline, C. D. Frith, and R. S. J. Frackowiak. Statistical parametric maps in functional imaging: A general linear approach. *HBM*, 2(4):189–210, 1995.

[8] U. Hasson, Y. Nir, I. Levy, G. Fuhrmann, and R. Malach. Intersubject synchronization of cortical activity during natural vision. *Science*, 303(1634), 2004.

[9] Y. Kamitani and F. Tong. Decoding the visual and subjective contents of the human brain. *Nature Neuroscience*, 8:679–685, 2005.

[10] J. Lafferty, A. McCallum, and F. Pereira. Conditional random fields: Probabilistic models for segmenting and labeling sequence data. In *ICML*, 2001.

[11] S. Lauritzen. *Graphical Models*. Oxford University Press, New York, 1996.

[12] N. K. Logothetis. The underpinnings of the bold functional magnetic resonance imaging signal. *The Journal of Neuroscience*, 23(10):3963–3971, 2003.

[13] T. Mitchell, R. Hutchinson, R. Niculescu, F.Pereira, X. Wang, M. Just, and S. Newman. Learning to decode cognitive states from brain images. *Machine Learning*, 57(1–2):145–175, 2004.

[14] E. Olivetti, D. Sona, and S. Veeramachaneni. Gaussian process regression and recurrent neural networks for fmri image classification. Abstract presented, 12th HBM meeting, Florence, Italy, 2006.

[15] T. P. Speed and H. T. Kiiveri. Gaussian Markov distributions over finite graphs. *Annals of Statistics*, 14.
